# Large Margin Learning of Upstream Scene Understanding Models

**Jun Zhu**[†]        **Li-Jia Li**[‡]        **Fei-Fei Li**[‡]        **Eric P. Xing**[†]

[†]`{junzhu,epxing}@cs.cmu.edu`  [‡]`{lijiali,feifeili}@cs.stanford.edu`

[†]School of Computer Science, Carnegie Mellon University, Pittsburgh, PA 15213

[‡]Department of Computer Science, Stanford University, Stanford, CA 94305

## Abstract

Upstream supervised topic models have been widely used for complicated scene understanding. However, existing maximum likelihood estimation (MLE) schemes can make the prediction model learning independent of latent topic discovery and result in an imbalanced prediction rule for scene classification. This paper presents a joint max-margin and max-likelihood learning method for upstream scene understanding models, in which latent topic discovery and prediction model estimation are closely coupled and well-balanced. The optimization problem is efficiently solved with a variational EM procedure, which iteratively solves an online loss-augmented SVM. We demonstrate the advantages of the large-margin approach on both an 8-category sports dataset and the 67-class MIT indoor scene dataset for scene categorization.

## 1   Introduction

Probabilistic topic models like the latent Dirichlet allocation (LDA) [5] have recently been applied to a number of computer vision tasks such as objection annotation and scene classification due to their ability to capture latent semantic compositions of natural images [22, 23, 9, 13]. One of the advocated advantages of such models is that they do not require "supervision" during training, which is arguably preferred over supervised learning that would necessitate extra cost. But with the increasing availability of free on-line information such as image tags, user ratings, etc., various forms of "side-information" that can potentially offer "free" supervision have led to a need for new models and training schemes that can make effective use of such information to achieve better results, such as more discriminative topic representations of image contents, and more accurate image classifiers.

The standard unsupervised LDA ignores the commonly available supervision information, and thus can discover a sub-optimal topic representation for prediction tasks. Extensions to supervised topic models which can explore side information for discovering predictive topic representations have been proposed, such as the sLDA [4, 25] and MedLDA [27]. A common characteristic of these models is that they are *downstream*, that is, the supervised response variables are *generated* from topic assignment variables. Another type of supervised topic models are the so-called *upstream* models, of which the response variables directly or indirectly *generate* latent topic variables. In contrast to downstream supervised topic models (dSTM), which are mainly designed by machine learning researchers, upstream supervised topic models (uSTM) are well-motivated from human vision and psychology research [18, 10] and have been widely used for scene understanding tasks. For example, in the recently developed scene understanding models [23, 13, 14, 8], complex scene images are modeled as a hierarchy of semantic concepts where the most top level corresponds to a *scene*, which can be represented as a set of latent objects likely to be found in a given scene. To learn an upstream scene model, maximum likelihood estimation (MLE) is the most common choice. However, MLE can make the prediction model estimation independent of latent topic discovery and result in an imbalanced prediction rule for scene classification, as we explain in Section 3.

In this paper, our goal is to address the weakness of MLE for learning upstream supervised topic models. Our approach is based on the max-margin principle for supervised learning which has shown great promise in many machine learning tasks, such as classification [21] and structured output prediction [24]. For the dSTM, max-margin training has been developed in MedLDA [27], which has achieved better prediction performance than MLE. In such downstream models, latent topic assignments are sufficient statistics for the prediction model and it is easy to define the max-margin constraints based on existing max-margin methods (e.g., SVM). However, for upstream supervised topic models, the discriminant function for prediction involves an intractable computation of posterior distributions, which makes the max-margin training more delicate.

Specifically, we present a joint max-margin and max-likelihood estimation method for learning upstream scene understanding models. By using a variational approximation to the posterior distribution of supervised variables (e.g., scene categories), our max-margin learning approach iterates between posterior probabilistic inference and max-margin parameter learning. The parameter learning solves an *online loss-augmented SVM*, which closely couples the prediction model estimation and latent topic discovery, and this close interplay results in a well-balanced prediction rule for scene categorization. Finally, we demonstrate the advantages of our max-margin approach on both the 8-category sports [13] and the 67-class MIT indoor scene [20] datasets. Empirical results show that max-margin learning can significantly improve the scene classification accuracy.

The paper is structured as follows. Sec. 2 presents a generic scene understanding model we will work on. Sec. 3 discusses the weakness of MLE in learning upstream models. Sec. 4 presents the max-margin learning approach. Sec. 5 presents empirical results and Sec. 6 concludes.

## 2   Joint Scene and Object Model: a Generic Running Example

In this section, we present a generic joint scene categorization and object annotation model, which will be used to demonstrate the large margin learning of upstream scene understanding models.

### 2.1   Image Representation

How should we represent a scene image? Friedman [10] pointed out that object recognition is critical in the recognition of a scene. While individual objects contribute to the recognition of visual scenes, human vision researchers Navon [18] and Biederman [2] also showed that people perform rapid global scene analysis before conducting more detailed local object analysis when recognizing scene images. To obtain a generic model, we represent a scene by using its global scene features and objects within it. We first segment an image $\mathcal{I}$ into a set of local regions $\{r_1, \cdots, r_N\}$. Each region is represented by three region features $R$ (i.e., color, location and texture) and a set of image patches $X$. These region features are represented as visual codewords. To describe detailed local information of objects, we partition each region into patches. For each patch, we extract the SIFT [16] features, which are insensitive to view-point and illumination changes. To model the global scene representation, we extract a set of global features $G$ [19]. In our dataset, we represent an image as a tuple $(\mathbf{r}, \mathbf{x}, \mathbf{g})$, where $\mathbf{r}$ denotes an instance of $R$, and likewise for $\mathbf{x}$ and $\mathbf{g}$.

### 2.2   The Joint Scene and Object Model

The model is shown in Fig. 1 (a). $S$ is the scene random variable, taking values from a finite set $\mathcal{S} = \{s_1, \cdots, s_{M_s}\}$. For an image, the distribution over scene categories depends on its global representation features $G$. Each scene is represented as a mixture over latent objects $O$ and the mixing weights are defined with a generalized linear model (GLM) parameterized by $\psi$. By using a normal prior on $\psi$, the scene model can capture the mutual correlations between different objects, similar to the correlated topic models (CTMs) [3]. Here, we assume that for different scenes, the objects have different distributions and correlations. Let $\mathbf{f}$ denote the vector of real-valued feature functions of $S$ and $G$, the generating procedure of an image is as follows:

1. Sample a scene category from a conditional scene model: $p(s|\mathbf{g}, \theta) = \frac{\exp(\theta^\top \mathbf{f}(\mathbf{g}, s))}{\sum_{s'} \exp(\theta^\top \mathbf{f}(\mathbf{g}, s'))}$.

2. Sample the parameters $\psi|s, \mu, \Sigma \sim \mathcal{N}(\mu_s, \Sigma_s)$.
3. For each region $n$
   (a) sample an object from: $p(o_n = k|\psi) = \frac{\exp(\psi_k)}{\sum_j \exp(\psi_j)}$.

   (b) sample $M_r$ (i.e., 3: color, location and texture) region features: $\mathbf{r}_{nm}|o_n, \beta \sim \text{Multi}(\beta_{mo_n})$.
   (c) sample $M_x$ image patches $\mathbf{x}_{nm}|o_n, \eta \sim \text{Multi}(\eta_{o_n})$.

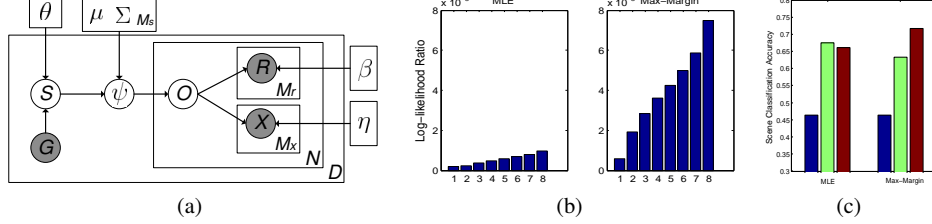

(a)                        (b)                        (c)

Figure 1: (a) a joint scene categorization and object annotation model with global features $G$; (b) average log-likelihood ratio $\log p(s|\mathbf{g}, \theta)/\mathcal{L}_{-\theta}$ under MLE and max-margin estimations, where the first bar is for true categories and the rest are for categories sorted based on their difference from the first one; (c) scene classification accuracy by using (Blue) $\mathcal{L}_{-\theta}$, (Green) $\log p(s|\mathbf{g}, \theta)$, and (Red) $\mathcal{L}_{-\theta} + \log p(s|\mathbf{g}, \theta)$ for prediction. Group 1 is for MLE and group 2 is for max-margin training.

The generative model defines a joint distribution

$$p(s, \psi, o, \mathbf{r}, \mathbf{x}|\mathbf{g}, \Theta) = p(s|\theta, \mathbf{g})p(\psi|\mu_s, \Sigma_s) \prod_{n=1}^{N} \Big( p(o_n|\psi) \prod_{m=1}^{M_r} p(\mathbf{r}_{nm}|o_n, \beta) \prod_{m=1}^{M_x} p(\mathbf{x}_{nm}|o_n, \eta) \Big),$$

where we have used $\Theta$ to denote all the unknown parameters $(\theta, \mu, \Sigma, \beta, \eta)$. From the joint distribution, we can make two types of predictions, namely scene classification and object annotation. For scene classification, we infer the maximum *a posteriori* prediction

$$\hat{s} \triangleq \arg\max_s p(s|\mathbf{g}, \mathbf{r}, \mathbf{x}) = \arg\max_s \log p(s, \mathbf{r}, \mathbf{x}|\mathbf{g}). \tag{1}$$

For object annotation, we can use the inferred latent representation of regions based on $p(o|\mathbf{g}, \mathbf{r}, \mathbf{x})$ and build a classifier to categorize regions into object classes, when some training examples with manually annotated objects are provided. Since collecting fully labeled images with annotated objects is difficult, upstream scene models are usually learned with partially labeled images for scene categorization, where only scene categories are provided and objects are treated as latent topics or themes [9]. In this paper, we focus on scene classification. Some empirical results on object annotation will be reported when labeled objects are available.

We use this joint model as a running example to demonstrate the basic principe of performing max-margin learning for the widely applied upstream scene understanding models because it is well-motivated, very generic and covers many other existing scene understanding models. For example, if we do not incorporate the global scene representation $G$, the joint model will be reduced to a model similar as [14, 6, 23]. Moreover, the generic joint model provides a good framework for studying the relative contributions of local object modeling and global scene representation, which has been shown to be useful for scene classification [20] and object detection [17] tasks.

## 3   Weak Coupling of MLE in Learning Upstream Scene Models

To learn an upstream scene model, the most commonly used method is the maximum likelihood estimation (MLE), such as in [23, 6, 14]. In this section, we discuss the weakness of MLE for learning upstream scene models and motivate the max-margin approach.

Let $\mathcal{D} = \{(\mathcal{I}_d, s_d)\}_{d=1}^{D}$ denote a set of partially labeled training images. The standard MLE obtains the optimum model parameters by maximizing the log-likelihood[1] $\sum_{d=1}^{D} \log p(s_d, \mathbf{r}_d, \mathbf{x}_d|\mathbf{g}_d, \Theta)$. By using the factorization of $p(s, \psi, o, \mathbf{r}, \mathbf{x}|\mathbf{g}, \Theta)$, MLE solves the following equivalent problem

$$\max_{\theta, \Theta_{-\theta}} \sum_d \big( \log p(s_d|\mathbf{g}_d, \theta) + L_{s_d, -\theta} \big), \tag{2}$$

where $L_{s_d, -\theta} \triangleq \log \int_\psi \sum_o p(\psi, o, \mathbf{r}_d, \mathbf{x}_d|s_d, \Theta) = \log p(\mathbf{r}_d, \mathbf{x}_d|s_d, \Theta)$ is the log-likelihood of image features given the scene class, and $\Theta_{-\theta}$ denotes all the parameters except $\theta$.

Since $L_{s, -\theta}$ does not depend on $\theta$, the MLE estimation of the conditional scene model is to solve

$$\max_{\theta} \sum_d \log p(s_d|\mathbf{g}_d, \theta), \tag{3}$$

which does not depend on the latent object model. This is inconsistent with the prediction rule (1) which does depend on both the conditional scene model (i.e., $p(s|\mathbf{g}, \theta)$) and the local object model.

This decoupling will result in an imbalanced combination between the conditional scene and object models for prediction, as we explain below.

We first present some details of the MLE method. For $\theta$, the problem (3) is an MLE estimation of a GLM, and it can be efficiently solved with gradient descent methods, such as quasi-Newton methods [15]. For $\Theta_{-\theta}$, since the likelihood $L_{s,-\theta}$ is intractable to compute, we apply variational methods to obtain an approximation. By introducing a variational distribution $q_s(\psi, o)$ to approximate the posterior $p(\psi, o|s, \mathbf{r}, \mathbf{x}, \Theta)$ and using the Jensen's inequality, we can derive a lower bound

$$L_{s,-\theta} \geq \mathbb{E}_{q_s}[\log p(\psi, o, \mathbf{r}, \mathbf{x}|s, \Theta)] + H(q_s) \triangleq \mathcal{L}_{-\theta}(q_s, \Theta), \tag{4}$$

where $H(q) = -\mathbb{E}_q[q]$ is the entropy. Then, the intractable prediction rule (1) can be approximated with the variational prediction rule

$$\hat{s} \triangleq \arg\max_{s,q_s} \left( \log p(s|\mathbf{g}, \theta) + \mathcal{L}_{-\theta}(q_s, \Theta) \right). \tag{5}$$

Maximizing $\sum_d \mathcal{L}_{-\theta}(q_{s_d}, \Theta)$ will lead to a closed form solution of $\Theta_{-\theta}$. See Appendix for the inference of $q_s$ as involved in the prediction rule (5) and the estimation of $\Theta_{-\theta}$.

Now, we examine the effects of the conditional scene model $p(s|\mathbf{g}, \theta)$ in making a prediction via the prediction rule (5). Fig. 1 (b-left) shows the relative importance of $\log p(s|\mathbf{g}, \theta)$ in the joint decision rule (5) on the sports dataset [13]. We can see that in MLE the conditional scene model plays a very weak role in making a prediction when it is combined with the object model, i.e., $\mathcal{L}_{-\theta}$. Therefore, as shown in Fig. 1 (c), although a simple logistic regression with global features (i.e., the green bar) can achieve a good accuracy, the accuracy of the prediction rule (5) that uses the joint likelihood bound (i.e, the red bar) is decreased due to the strong effect of the potentially bad prediction rule based on $\mathcal{L}_{-\theta}$ (i.e., the blue bar), which only considers local image features.

In contrast, as shown in Fig. 1 (b-right), in the max-margin approach to be presented, the conditional scene model plays a much more influential role in making a prediction via the rule (5). This results in a better balanced combination between the scene and the object models. The strong coupling is due to solving an *online loss-augmented SVM*, as we explain below. Note that we are not claiming any weakness of MLE in general. All our discussions are concentrated on learning upstream supervised topic models, as generically represented by the model in Fig. 1.

## 4 Max-Margin Training

Now, we present the max-margin method for learning upstream scene understanding models.

### 4.1 Problem Definition

For the predictive rule (1), we use $F(s, \mathbf{g}, \mathbf{r}, \mathbf{x}; \Theta) \triangleq \log p(s|\mathbf{g}, \mathbf{r}, \mathbf{x}, \Theta)$ to denote the *discriminant* function, which is more complicated than the commonly chosen linear form, in the sense we will explain shortly. In the same spirit of max-margin classifiers (e.g., SVMs), we define the hinge loss of the prediction rule (1) on $\mathcal{D}$ as

$$\mathcal{R}_{hinge}(\Theta) = \frac{1}{D} \sum_d \max_s [\Delta \ell_d(s) - \Delta F_d(s; \Theta)],$$

where $\Delta \ell_d(s)$ is a loss function (e.g., 0/1 loss), and $\Delta F_d(s; \Theta) = F(s_d, \mathbf{g}_d, \mathbf{r}_d, \mathbf{x}_d; \Theta) - F(s, \mathbf{g}_d, \mathbf{r}_d, \mathbf{x}_d; \Theta)$ is the margin favored by the true category $s_d$ over any other category $s$.

The problem with the above definition is that exactly computing the posterior distribution $p(s|\mathbf{g}, \mathbf{r}, \mathbf{x}, \Theta)$ is intractable. As in MLE, we use a variational distribution $q_s$ to approximate it. By using the Bayes's rule and the variational bound in Eq. (4), we can lower bound the log-likelihood

$$\log p(s|\mathbf{g}, \mathbf{r}, \mathbf{x}, \Theta) = \log p(s, \mathbf{r}, \mathbf{x}|\mathbf{g}, \Theta) - \log p(\mathbf{r}, \mathbf{x}|\mathbf{g}, \Theta) \geq \log p(s|\mathbf{g}, \theta) + \mathcal{L}_{-\theta}(q_s, \Theta) - c, \tag{6}$$

where $c = \log p(\mathbf{r}, \mathbf{x}|\mathbf{g}, \Theta)$. Without causing ambiguity, we will use $\mathcal{L}_{-\theta}(q_s)$ without $\Theta$. Since we need to make some assumptions about $q_s$, the equality in (6) usually does not hold. Therefore, the tightest lower bound is an approximation of the intractable discriminant function

$$F(s, \mathbf{g}, \mathbf{r}, \mathbf{x}; \Theta) \approx \log p(s|\mathbf{g}, \theta) + \max_{q_s} \mathcal{L}_{-\theta}(q_s) - c. \tag{7}$$

Then, the margin is $\Delta F_d(s; \Theta) = \theta^\top \Delta \mathbf{f}_d(s) + \max_{q_{s_d}} \mathcal{L}_{-\theta}(q_{s_d}) - \max_{q_s} \mathcal{L}_{-\theta}(q_s)$, of which the linear term is the same as that in a linear SVM [7] and the difference between two variational bounds causes the topic discovery to bias the learning of the scene classification model, as we shall see.

Using the variational discriminant function in Eq. (7) and applying the principle of *regularized empirical risk minimization*, we define the max-margin learning of the joint scene and object model as solving

$$\min_{\Theta} \; \Omega(\Theta) + \lambda \sum_d (-\max_{q_{s_d}} \mathcal{L}_{-\theta}(q_{s_d})) + C\mathcal{R}_{hinge}(\Theta), \tag{8}$$

where $\Omega(\Theta)$ is a regularizer of the parameters. Here, we define $\Omega(\Theta) \triangleq \frac{1}{2}\|\theta\|_2^2$. For the normal mean $\mu_s$ or covariance matrix $\Sigma_s$, a similar $\ell_2$-norm or Frobenius norm can be used without changing our algorithm. The free parameters $\lambda$ and $C$ are positive and tradeoff the classification loss and the data likelihood. When $\lambda \to \infty$, the problem (8) reduces to the standard MLE of the joint scene model with a fixed uniform prior on scene classes. Moreover, we can see the difference from the standard MLE (2). Here, we minimize a hinge loss, which is defined on the joint prediction rule, while MLE minimizes the log-likelihood loss $\log p(s_d|\mathbf{g}_d, \theta)$, which does not depend on the latent object model. Therefore, our approach can be expected to achieve a closer dependence between the conditional scene model and the latent object model. More insights will be provided in the next section.

## 4.2 Solving the Optimization Problem

The problem (8) is generally hard to solve because the model parameters and variational distributions are strongly coupled. Therefore, we develop a natural iterative procedure that estimates the parameters $\Theta$ and performs posterior inference alternatively. The intuition is that by fixing one part (e.g., $q_s$) the other part (e.g., $\Theta$) can be efficiently done. Specifically, using the definitions, we rewrite the problem (8) as a min-max optimization problem

$$\min_{\Theta, \{q_{s_d}\}} \max_{\{s, q_s\}} \left( \frac{1}{2}\|\Theta\|_2^2 - (\lambda + C)\sum_d \mathcal{L}_{-\theta}(q_{s_d}) + C\sum_d [-\theta^\top \Delta\mathbf{f}_d(s) + \Delta\ell_d(s) + \mathcal{L}_{-\theta}(q_s)] \right), \tag{9}$$

where the factor $1/D$ in $\mathcal{R}_{hinge}$ is absorbed in the constant $C$. This min-max problem can be approximately solved with an iterative procedure. First, we infer the optimal variational posterior[2] $q_s^\star = \arg\max_{q_s} \mathcal{L}_{-\theta}(q_s)$ for each $s$ and each training image. Then, we solve

$$\min_{\Theta, \{q_{s_d}\}} \left( \frac{1}{2}\|\Theta\|_2^2 - (\lambda + C)\sum_d \mathcal{L}_{-\theta}(q_{s_d}) + C\sum_d \max_s [-\theta^\top \Delta\mathbf{f}_d(s) + \Delta\ell_d(s) + \mathcal{L}_{-\theta}(q_s^\star)] \right),$$

For this sub-step, again, we apply an alterative procedure to solve the minimization problem over $\Theta$ and $q_{s_d}$. We first infer the optimal variational posterior $q_{s_d}^\star = \arg\max_{q_{s_d}} \mathcal{L}_{-\theta}(q_{s_d})$, and then we estimate the parameters by solving the following problem

$$\min_{\Theta} \left( \frac{1}{2}\|\Theta\|_2^2 - (\lambda + C)\sum_d \mathcal{L}_{-\theta}(q_{s_d}^\star) + C\sum_d \max_s [-\theta^\top \Delta\mathbf{f}_d(s) + \Delta\ell_d(s) + \mathcal{L}_{-\theta}(q_s^\star)] \right), \tag{10}$$

Since inferring $q_{s_d}^\star$ is included in the step of inferring $q_s^\star$ ($\forall s$), the algorithm can be summarized as a two-step EM-procedure that iteratively performs posterior inference of $q_s$ and max-margin parameter estimation. Another way to understand this iterative procedure is from the definitions. The first step of inferring $q_s^\star$ is to compute the discriminant function $F$ under the current model. Then, we update the model parameters $\Theta$ by solving a large-margin learning problem. For brevity, we present the parameter estimation only. The posterior inference is detailed in Appendix A.1.

**Parameter Estimation**: This step can be done with an alternating minimization procedure. For the Gaussian parameters $(\mu, \Sigma)$ and multinomial parameters $(\eta, \beta)$, the estimation can be written in a closed-form as in a standard MLE of CTMs [3] by using a loss-augmented prediction of $s$. For brevity, we defer the details to the Appendix A.2. Now, we present the step of estimating $\theta$, which illustrates the essential difference between the large-margin approach and the standard MLE. Specifically, the optimum solution of $\theta$ is obtained by solving the sub-problem[3]

$$\min_{\theta} \frac{1}{2}\|\theta\|_2^2 + C\sum_d \left( \max_s [\theta^\top \mathbf{f}(\mathbf{g}_d, s) + \Delta\ell_d(s) + \mathcal{L}_{-\theta}(q_s^\star)] - [\theta^\top \mathbf{f}(\mathbf{g}_d, s_d) + \mathcal{L}_{-\theta}(q_{s_d}^\star)] \right),$$

which is equivalent to a constrained problem by introducing a set of non-negative slack variables $\xi$

$$\min_{\theta, \xi} \; \frac{1}{2}\|\theta\|_2^2 + C\sum_{d=1}^{D} \xi_d \quad \text{s.t.: } \theta^\top \Delta\mathbf{f}_d(s) + [\mathcal{L}_{-\theta}(q_{s_d}^\star) - \mathcal{L}_{-\theta}(q_s^\star)] \geq \Delta\ell_d(s) - \xi_d, \forall d, s. \tag{11}$$

The constrained optimization problem is similar to that of a linear SVM [7]. However, the difference is that we have the additional term

$$\Delta \mathcal{L}_d^\star(s) \triangleq \mathcal{L}_{-\theta}(q_{s_d}^\star) - \mathcal{L}_{-\theta}(q_s^\star).$$

This term indicates that the estimation of the scene classification model is influenced by the topic discovery procedure, which finds an optimum posterior distribution $q^\star$. If $\Delta \mathcal{L}_d^\star(s) < 0$, $s \neq s_d$, which means it is very likely that a wrong scene $s$ explains the image content better than the true scene $s_d$, then the term $\Delta \mathcal{L}_d^\star(s)$ acts in a role of augmenting the linear decision boundary $\theta$ to make a correct prediction on this image by using the prediction rule (5). If $\Delta \mathcal{L}_d^\star(s) > 0$, which means the true scene can explain the image content better than $s$, then the linear decision boundary can be slightly relaxed. If we move the additional term to the right hand side, the problem (11) is to learn a linear SVM, but with an online updated loss function $\Delta \ell_d(s) - \Delta \mathcal{L}_d^\star(s)$. We call this SVM an *online loss-augmented SVM*. Solving the loss-augmented SVM will result in an amplified influence of the scene classification model in the joint predictive rule (5) as shown in Fig. 1 (b).

## 5   Experiments

Now, we present empirical evaluation of our approach on the sports [13] and MIT indoor scene [20] datasets. Our goal is to demonstrate the advantages of the max-margin method over the MLE for learning upstream scene models with or without global features. Although the model in Fig. 1 can also be used for object annotation, we report the performance on scene categorization only, which is our main focus in this paper. For object annotation, which requires additional human annotated examples of objects, some preliminary results are reported in the Appendix due to space limitation.

### 5.1   Datasets and Features

The sports data contain 1574 diverse scene images from 8 categories, as listed in Fig. 2 with example images. The indoor scene dataset [20] contains 15620 scene images from 67 categories as listed in Table 2. We use the method [1] to segment these images into small regions based on color, brightness and texture homogeneity. For each region, we extract color, texture and location features, and quantize them into 30, 50 and 120 codewords, respectively. Similarly, the SIFT features extracted from the small patches within each region are quantized into 300 SIFT codewords. We use the gist features [19] as one example of global features. Extension to include other global features, such as SIFT sparse codes [26], can be directly done without changing the model or the algorithm.

### 5.2   Models

For the upstream scene model as in Fig. 1, we compare the max-margin learning with the MLE method, and we denote the scene models trained with max-margin training and MLE by *MM-Scene* and *MLE-Scene*, respectively. For both methods, we evaluate the effectiveness of global features, and we denote the scene models without global features by *MM-Scene-NG* and *MLE-Scene-NG*, respectively. Since our main goal in this paper is to demonstrate the advantages of max-margin learning in upstream supervised topic models, rather than dominance of such models over all others, we just compare with one example of downstream models–the multi-class sLDA (Multi-sLDA) [25]. Systematical comparison with other methods, including DiscLDA [12] and MedLDA [27], is deferred to a full version. For the downstream Multi-sLDA, the image-wise scene category variable $S$ is generated from latent object variables $O$ via a softmax function. For this downstream model, the parameter estimation can be done with MLE as detailed in [25].

Finally, to show the usefulness of the object model in scene categorization, we also compare with the margin-based multi-class SVM [7] and likelihood-based logistic regression for scene classification based on the global features. For the SVM, we use the software SVM$^{multiclass}$ [4], which implements a fast cutting-plane algorithm [11] to do parameter learning. We use the same software with slight changes to learn the loss-augmented SVM in our max-margin method.

### 5.3   Scene Categorization on the 8-Class Sports Dataset

We partition the dataset equally into training and testing data. For all the models except SVM and logistic regression, we run 5 times with random initialization of the topic parameters (e.g., $\beta$ and $\eta$).

[4]http://svmlight.joachims.org/svm_multiclass.html

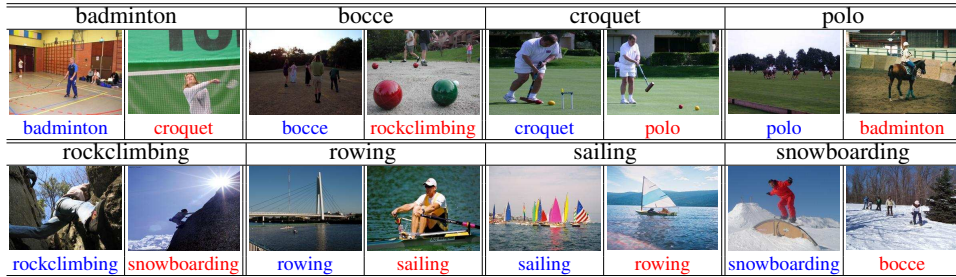

| badminton | | bocce | | croquet | | polo | |
|---|---|---|---|---|---|---|---|
| badminton | croquet | bocce | rockclimbing | croquet | polo | polo | badminton |
| rockclimbing | | rowing | | sailing | | snowboarding | |
| rockclimbing | snowboarding | rowing | sailing | sailing | rowing | snowboarding | bocce |

Figure 2: Example images from each category in the sports dataset with predicted scene classes, where the predictions in blue are correct while red ones are wrong predictions.

The average overall accuracy of scene categorization on 8 categories and its standard deviation are shown in Fig. 3. The result of logistic regression is shown in the left green bar in Fig. 1 (c). We also show the confusion matrix of the max-margin scene model with 100 latent topics in Table 1, and example images from each category are shown in Fig. 2 with predicted labels. Overall, the max-margin scene model with global features achieves significant improvements as compared to all other approaches we have tested. Interestingly, although we provide only scene categories as supervised information during training, our best performance with global features is close to that reported in [13], where

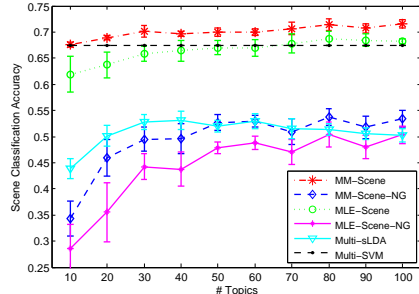

Figure 3: Classification accuracy of different models with respect to the number of topics.

additional supervision of objects is used. The outstanding performance of the max-margin method for scene classification can be understood from the following aspects.

**Max-margin training**: from the comparison of the max-margin approach with the standard MLE in both cases of using global features and not using global features, we can see that the max-margin learning can improve the performance dramatically, especially when the scene model uses global features (about 3 percent). This is due to the well-balanced prediction rule achieved by the max-margin method, as we have explained in Section 3.

**Global features**: from the comparison between the scene models with and without global features, we can see that using the gist features can significantly (about 8 percent) improve the scene categorization accuracy in both MLE and max-margin training. We also did some preliminary experiments on the SIFT sparse codes feature [26], which are a bit more expensive to extract. By using both gist and sparse codes features, we can achieve dramatic improvements in both max-margin and MLE methods. Specifically, the max-margin scene model achieves an accuracy of about 0.83 in scene classification, and the likelihood-based model obtains an accuracy of about 0.80.

**Object modeling**: the superior performance of the max-margin learned *MM-scene* model comparing to the SVM and logistic regression (See the left green bar of Fig. 1 (c)), which use global features only, indicates that modeling objects can facilitate scene categorization. This is because the scene classification model is influenced by the latent object modeling through the term $\Delta \mathcal{L}_d^\star(s)$, which can improve the decision boundary of a standard linear SVM for those images that have negative scores of $\Delta \mathcal{L}_d^\star(s)$, as we have discussed in the online loss-augmented SVM. However, object modeling does not improve the classification accuracy and sometimes it can even be harmful when the scene model is learned with the standard MLE. This is because the object model (using the state-of-the-art representation) (e.g., MM-MLE-NG) alone performs much worse than global feature models (e.g., logistic regression), as shown in Fig. 1 and Fig. 3, and the standard MLE learns an imbalanced prediction rule, as we have analyzed in Section 3. Given that the state-of-the-art object model is not good, it is very encouraging to see that we can still obtain positive improvements by using the closely coupled and well-balanced max-margin learning. These results indicate that further improvements can be expected by improving the local object model, e.g., by incorporating rich features.

We also compare with the theme model [9], which is for scene categorization only. The theme model uses a different image representation, where each image is a vector of image patch codewords. The theme model achieves about 0.65 in classification accuracy, lower than that of MM-Scene.

Table 1: Confusion matrix for 100-topic MM-Scene on the sports dataset.

| 0.717 | badminton | bocce | croquet | polo | rock-climbing | rowing | sailing | snow-boarding |
|---|---|---|---|---|---|---|---|---|
| badminton | **0.768** | 0.051 | 0.051 | 0.081 | 0.020 | 0.020 | 0.000 | 0.010 |
| bocce | 0.043 | **0.333** | 0.275 | 0.145 | 0.087 | 0.058 | 0.014 | 0.043 |
| croquet | 0.025 | 0.144 | **0.669** | 0.093 | 0.025 | 0.025 | 0.008 | 0.008 |
| polo | 0.220 | 0.055 | 0.099 | **0.516** | 0.022 | 0.022 | 0.011 | 0.055 |
| rockclimbing | 0.000 | 0.010 | 0.021 | 0.000 | **0.845** | 0.031 | 0.010 | 0.082 |
| rowing | 0.008 | 0.008 | 0.008 | 0.008 | 0.024 | **0.912** | 0.016 | 0.016 |
| sailing | 0.011 | 0.021 | 0.000 | 0.021 | 0.011 | 0.053 | **0.884** | 0.000 |
| snowboarding | 0.011 | 0.021 | 0.032 | 0.095 | 0.084 | 0.053 | 0.063 | **0.642** |

Table 2: The 67 indoor categories sorted by classification accuracy by 70-topic MM-Scene.

| | | | |
|---|---|---|---|
| buffet 0.85 | lobby 0.40 | stairscase 0.25 | hospitalroom 0.10 |
| green house 0.84 | prison cell 0.39 | studiomusic 0.24 | kindergarden 0.10 |
| cloister 0.71 | casino 0.36 | children room 0.21 | laundromat 0.10 |
| inside bus 0.61 | dining room 0.35 | garage 0.20 | office 0.10 |
| movie theater 0.60 | kitchen 0.35 | gym 0.20 | restaurant kitchen 0.09 |
| poolinside 0.59 | winecellar 0.34 | hairsalon 0.20 | shoeshop 0.09 |
| church inside 0.56 | library 0.31 | livingroom 0.20 | videostore 0.08 |
| classroom 0.55 | tv studio 0.30 | operating room 0.20 | airport inside 0.07 |
| concert hall 0.55 | warehouse 0.29 | pantry 0.20 | bar 0.06 |
| corridor 0.55 | batchroom 0.26 | subway 0.20 | deli 0.06 |
| florist 0.55 | bookstore 0.25 | toystore 0.19 | jewelleryshop 0.06 |
| trainstation 0.54 | computerroom 0.25 | artstudio 0.14 | laboratorywet 0.05 |
| closet 0.51 | dentaloffice 0.25 | fastfood restaurant 0.13 | locker room 0.05 |
| elevator 0.49 | grocerystore 0.25 | auditorium 0.12 | museum 0.05 |
| nursery 0.44 | inside subway 0.25 | bakery 0.11 | restaurant 0.05 |
| bowling 0.41 | mall 0.25 | bedroom 0.11 | waitingroom 0.04 |
| gameroom 0.40 | meeting room 0.25 | clothingstore 0.10 | |

Finally, we examine the influence of the loss function $\Delta\ell_d(s)$ on the performance of the max-margin scene model. As we can see in problem (11), the loss function $\Delta\ell_d(s)$ is another important factor that influences the estimation of $\theta$ and its relative importance in the prediction rule (5). Here, we use the $0/\ell$-loss function, that is, $\Delta\ell_d(s) = \ell$ if $s \neq s_d$; otherwise 0. Fig. 4 shows the performance of the 100-topic MM-Scene model when using different loss functions. When $\ell$ is set between 10 and 20, the MM-Scene method stably achieves the best performance. The above results in Fig. 3 and Table 1 are achieved with $\ell$ selected from 5 to 40 with cross-validation during training.

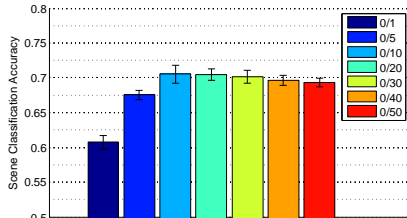

Figure 4: Classification accuracy of MM-Scene with different loss functions $\Delta\ell_d(s)$.

## 5.4 Scene Categorization on the 67-Class MIT Indoor Scene Dataset

The MIT indoor dataset [20] contains complex scene images from 67 categories. We use the same training and testing dataset as in [20], in which each category has about 80 images for training and about 20 images for testing. We compare the joint scene model with SVM, logistic regression (LR), and the prototype-based methods [20]. Both the SVM and LR are based on the global gist features only. For the joint scene model, we set the number of latent topics at 70. The overall performance of different methods are shown in Fig. 5 and the classification accuracy of each class is shown in Table 2. For the prototype-based methods, we cite the results from [20]. We can see that the joint scene

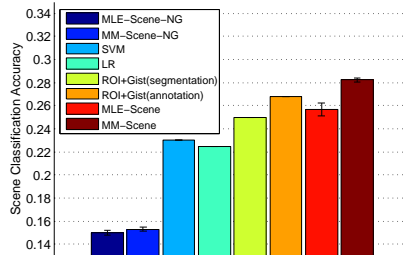

Figure 5: Classification accuracy on the 67-class MIT indoor dataset.

model (both MLE-Scene and MM-Scene) significantly outperforms SVM and LR that use global features only. The likelihood-based MLE-Scene slightly outperforms the ROI-Gist(segmentation), which uses both the global gist features and local region-of-interest (ROI) features extracted from automatically segmented regions [20]. By using max-margin training, the joint scene model (i.e., MM-Scene) achieves significant improvements compared to MLE-Scene. Moreover, the margin-based MM-Scene, which uses automatically segmented regions to extract features, outperforms the ROI-Gist(annotation) method that uses *human annotated* interested regions.

## 6 Conclusions

In this paper, we address the weak coupling problem of the commonly used maximum likelihood estimation in learning upstream scene understanding models by presenting a joint maximum margin and maximum likelihood learning method. The proposed approach achieves a close interplay between the prediction model estimation and latent topic discovery, and thereby a well-balanced prediction rule. The optimization problem is efficiently solved with a variational EM procedure, which iteratively learns an online loss-augmented SVM. Finally, we demonstrate the advantages of max-margin training and the effectiveness of using global features in scene understanding on both an 8-category sports dataset and the 67-class MIT indoor scene data.

## Acknowledgements

J.Z and E.P.X are supported by ONR N000140910758, NSF IIS-0713379, NSF Career DBI-0546594, and an Alfred P. Sloan Research Fellowship to E.P.X. L.F-F is partially supported by an NSF CAREER grant (IIS-0845230), a Google research award, and a Microsoft Research Fellowship. We also would like to thank Olga Russakovsky for helpful comments.

## Footnotes

[1]The conditional likelihood estimation can avoid this problem to some extend, but it has not been studied, to the best of our knowledge.

[2]To retain an accurate large-margin criterion for estimating model parameters (especially $\theta$), we do not perform the maximization over $s$ at this step.

[3]The constant (w.r.t. $\theta$) term $-C\sum_d \mathcal{L}_{-\theta}(q_{s_d}^\star)$ is kept for easy explanation. It won't change the estimation.

## References

[1] P. Arbeláez and L. Cohen. Constrained image segmentation from hierarchical boundaries. In *CVPR*, 2008.

[2] I. Biederman. On the semantics of a glance at a scene. *Perceptual Organization*, 213–253, 1981.

[3] D. Blei and J. Lafferty. Correlated topic models. In *NIPS*, 2006.

[4] D. Blei and J.D. McAuliffe. Supervised topic models. In *NIPS*, 2007.

[5] D. Blei, A. Ng, and M. Jordan. Latent Dirichlet allocation. *JMLR*, (3):993–1022, 2003.

[6] L.-L. Cao and L. Fei-Fei. Spatially coherent latent topic model for concurrent segmentation and classification of objects and scenes. In *ICCV*, 2007.

[7] K. Crammer and Y. Singer. On the algorithmic implementation of multiclass kernel-based vector machines. *JMLR*, (2):265–292, 2001.

[8] L. Du, L. Ren, D. Dunson, and L. Carin. A bayesian model for simultaneous image cluster, annotation and object segmentation. In *NIPS*, 2009.

[9] L. Fei-Fei and P. Perona. A bayesian hierarchical model for learning natural scene categories. In *CVPR*, 2005.

[10] A. Friedman. Framing pictures: The role of knowledge in automatized encoding and memory for gist. *Journal of Experimental Psychology: General*, 108(3):316–355, 1979.

[11] T. Joachims, T. Finley, and C.-N. Yu. Cutting-plane training of structural SVMs. *Machine Learning*, 77(1):27–59, 2009.

[12] S. Lacoste-Jullien, F. Sha, and M. Jordan. DiscLDA: Discriminative learning for dimensionality reduction and classification. In *NIPS*, 2008.

[13] L.-J. Li and L. Fei-Fei. What, where and who? classifying events by scene and object recognition. In *CVPR*, 2007.

[14] L.-J. Li, R. Socher, and L. Fei-Fei. Towards total scene understanding: Classification, annotation and segmentation in an automatic framework. In *CVPR*, 2009.

[15] D.C. Liu and J. Nocedal. On the limited memory BFGS method for large scale optimization. *Mathematical Programming*, (45):503–528, 1989.

[16] D.G. Lowe. Object recognition from local scale-invariant features. In *ICCV*, 1999.

[17] K. Murphy, A. Torralba, and W. Freeman. Using the forest to see the trees: A graphical model relating features, objects, and scenes. In *NIPS*, 2003.

[18] D. Navon. Forest before trees: The precedence of global features in visual perception. *Perception and Psychophysics*, 5:197–200, 1969.

[19] A. Oliva and A. Torralba. Modeling the shape of the scene: a holistic representation of the spatial envelope. *IJCV*, 42(3):145–175, 2001.

[20] A. Quattoni and A. Torralba. Recognizing indoor scenes. In *CVPR*, 2009.

[21] B. Schölkopf and A. Smola. *Learning with Kernels: Support Vector Machines, Regularization, Optimization, and Beyond.* MIT Press, 2001.

[22] J. Sivic, B.C. Russell, A. Efros, A. Zisserman, and W.T. Freeman. Discovering objects and their locatioins in images. In *ICCV*, 2005.

[23] E. Sudderth, A. Torralba, W. Freeman, and A. Willsky. Learning hierarchical models of scenes, objects, and parts. In *CVPR*, 2005.

[24] B. Taskar, C. Guestrin, and D. Koller. Max-margin Markov networks. In *NIPS*, 2003.

[25] C. Wang, D. Blei, and L. Fei-Fei. Simultaneous image classification and annotation. In *CVPR*, 2009.

[26] J. Yang, K. Yu, Y. Gong, and T. Huang. Linear spatial pyramid matching using sparse coding forimage classification. In *CVPR*, 2009.

[27] J. Zhu, A. Ahmed, and E.P. Xing. MedLDA: Maximum margin supervised topic models for regression and classification. In *ICML*, 2009.

